# Joint Tracking of Pose, Expression, and Texture using Conditionally Gaussian Filters

**Tim K. Marks**      **John Hershey**
Department of Cognitive Science
University of California San Diego
La Jolla, CA 92093-0515
tkmarks@cogsci.ucsd.edu
hershey@microsoft.com

**J. Cooper Roddey**    **Javier R. Movellan**
Institute for Neural Computation
University of California San Diego
La Jolla, CA 92093-0523
cooper@sccn.ucsd.edu
movellan@mplab.ucsd.edu

## Abstract

We present a generative model and stochastic filtering algorithm for simultaneous tracking of 3D position and orientation, non-rigid motion, object texture, and background texture using a single camera. We show that the solution to this problem is formally equivalent to stochastic filtering of conditionally Gaussian processes, a problem for which well known approaches exist [3, 8]. We propose an approach based on Monte Carlo sampling of the nonlinear component of the process (object motion) and exact filtering of the object and background textures given the sampled motion. The smoothness of image sequences in time and space is exploited by using Laplace's method to generate proposal distributions for importance sampling [7]. The resulting inference algorithm encompasses both optic flow and template-based tracking as special cases, and elucidates the conditions under which these methods are optimal. We demonstrate an application of the system to 3D non-rigid face tracking.

## 1  Background

Recent algorithms track morphable objects by solving optic flow equations, subject to the constraint that the tracked points belong to an object whose non-rigid deformations are linear combinations of a set of basic shapes [10, 2, 11]. These algorithms require precise initialization of the object pose and tend to drift out of alignment on long video sequences. We present *G-flow*, a generative model and stochastic filtering formulation of tracking that address the problems of initialization and error recovery in a principled manner.

We define a non-rigid object by the 3D locations of $n$ vertices. The object is a linear combination of $k$ fixed morph bases, with coefficients $c = [c_1, c_2, \cdots, c_k]^T$. The fixed $3 \times k$ matrix $h_i$ contains the position of the $i$th vertex in all $k$ morph bases. The transformation from object-centered to image coordinates consists of a rotation, weak perspective projection, and translation. Thus $x_i$, the 2D location of the $i$th vertex on the image plane, is

$$x_i = grh_ic + l, \tag{1}$$

where $r$ is the $3 \times 3$ rotation matrix, $l$ is the $2 \times 1$ translation vector, and $g = \left[\begin{smallmatrix} 1 & 0 & 0 \\ 0 & 1 & 0 \end{smallmatrix}\right]$ is the projection matrix. The object *pose*, $u_t$, comprises both the rigid motion parameters and the morph parameters at time $t$:

$$u_t = \{r(t), l(t), c(t)\}. \tag{2}$$

## 1.1 Optic flow

Let $y_t$ represent the current image, and let $x_i(u_t)$ index the image pixel that is rendered by the $i$th object vertex when the object assumes pose $u_t$. Suppose that we know $u_{t-1}$, the pose at time $t-1$, and we want to find $u_t$, the pose at time $t$. This problem can be solved by minimizing the following form with respect to $u_t$:

$$\hat{u}_t = \operatorname*{argmin}_{u_t} \frac{1}{2} \sum_{i=1}^{n} \left[y_t(x_i(u_t)) - y_{t-1}(x_i(u_{t-1}))\right]^2. \tag{3}$$

In the special case in which the $x_i(u_t)$ are neighboring points that move with the same 2D displacement, this reduces to the standard Lucas-Kanade optic flow algorithm [9, 1]. Recent work [10, 2, 11] has shown that in the general case, this optimization problem can be solved efficiently using the Gauss-Newton method. We will take advantage of this fact to develop an efficient stochastic inference algorithm within the framework of G-flow.

**Notational conventions**   Unless otherwise stated, capital letters are used for random variables, small letters for specific values taken by random variables, and Greek letters for fixed model parameters. Subscripted colons indicate sequences: e.g., $X_{1:t} = X_1 \cdots X_t$. The term $I_n$ stands for the $n \times n$ identity matrix, $E$ for expected value, $Var$ for the covariance matrix, and $Var^{-1}$ for the inverse of the covariance matrix (precision matrix).

## 2   The Generative Model for G-Flow

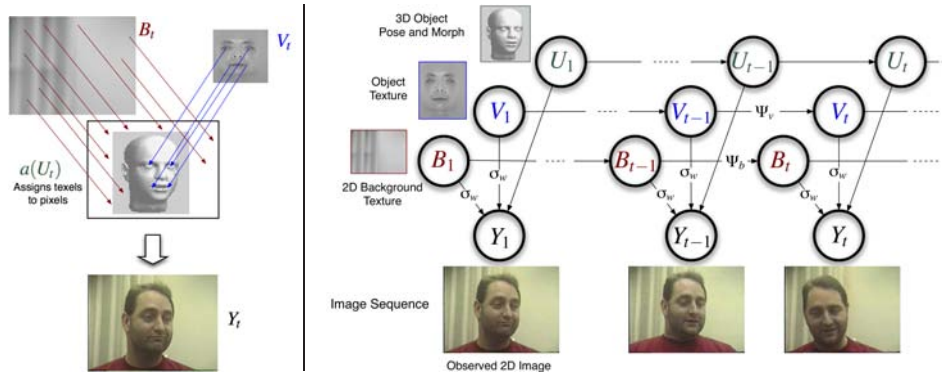

Figure 1: **Left:** $a(U_t)$ determines which texel (color at a vertex of the object model or a pixel of the background model) is responsible for rendering each image pixel. **Right:** G-flow video generation model: At time $t$, the object's 3D pose, $U_t$, is used to project the object texture, $V_t$, into 2D. This projection is combined with the background texture, $B_t$, to generate the observed image, $Y_t$.

We model the image sequence $Y$ as a stochastic process generated by three hidden causes, $U$, $V$, and $B$, as shown in the graphical model (Figure 1, right). The $m \times 1$ random vector $Y_t$ represents the $m$-pixel image at time $t$. The $n \times 1$ random vector $V_t$ and the $m \times 1$ random vector $B_t$ represent the $n$-texel object texture and the $m$-texel background texture, respectively. As illustrated in Figure 1, left, the object pose, $U_t$, determines onto which image pixels the object and background texels project at time $t$. This is formulated using the projection function $a(U_t)$. For a given pose, $u_t$, the projection $a(u_t)$ is a block matrix, $a(u_t) \stackrel{\text{def}}{=} \left[\ a_v(u_t) \ \vdots \ a_b(u_t)\ \right]$. Here $a_v(u_t)$, the object projection function, is an $m \times n$ matrix of 0s and 1s that tells onto which image pixel each object vertex projects; e.g., a 1 at row $j$, column $i$ it means that the $i$th object point projects onto image pixel $j$. Matrix $a_b$ plays the same role for background pixels. Assuming the foreground mapping is one-to-one, we let $a_b = I_m - a_v(u_t)a_v(u_t)^T$, expressing the simple occlusion constraint that every

image pixel is rendered by object or background, but not both. In the G-flow generative model:

$$Y_t = a(U_t)\begin{pmatrix} V_t \\ B_t \end{pmatrix} + W_t \qquad W_t \sim N(0, \sigma_w I_m), \qquad \sigma_w > 0$$

$$U_t \sim p(u_t \mid u_{t-1})$$
$$V_t = V_{t-1} + Z_{t-1}^v \qquad\qquad Z_{t-1}^v \sim N(0, \Psi_v), \qquad \Psi_v \text{ is diagonal}$$
$$B_t = B_{t-1} + Z_{t-1}^b \qquad\qquad Z_{t-1}^b \sim N(0, \Psi_b), \qquad \Psi_b \text{ is diagonal}$$

(4)

where $p(u_t \mid u_{t-1})$ is the pose transition distribution, and $Z^v, Z^b, W$ are independent of each other, of the initial conditions, and over time. The form of the pose distribution is left unspecified since the algorithm proposed here does not require the pose distribution or the pose dynamics to be Gaussian. For the initial conditions, we require that the variance of $V_1$ and the variance of $B_1$ are both diagonal.

Non-rigid 3D tracking is a difficult nonlinear filtering problem because changing the pose has a nonlinear effect on the image pixels. Fortunately, the problem has a rich structure that we can exploit: under the G-flow model, video generation is a conditionally Gaussian process [3, 6, 4, 5]. If the specific values taken by the pose sequence, $u_{1:t}$, were known, then the texture processes, $V$ and $B$, and the image process, $Y$, would be jointly Gaussian. This suggests the following scheme: we could use particle filtering to obtain a distribution of pose experts (each expert corresponds to a highly probable sample of pose, $u_{1:t}$). For each expert we could then use Kalman filtering equations to infer the posterior distribution of texture given the observed images. This method is known in the statistics community as a Monte Carlo filtering solution for conditionally Gaussian processes [3, 4], and in the machine learning community as Rao-Blackwellized particle filtering [6, 5]. We found that in addition to Rao-Blackwellization, it was also critical to use Laplace's method to generate the proposal distributions for importance sampling [7]. In the context of G-flow, we accomplished this by performing an optic flow-like optimization, using an efficient algorithm similar to those in [10, 2].

## 3  Inference

Our goal is to find an expression for the filtering distribution, $p(u_t, v_t, b_t \mid y_{1:t})$. Using the law of total probability, we have the following equation for the filtering distribution:

$$p(u_t, v_t, b_t \mid y_{1:t}) = \int \underbrace{p(u_t, v_t, b_t \mid u_{1:t-1}, y_{1:t})}_{\substack{\textbf{Opinion} \\ \text{of expert}}} \underbrace{p(u_{1:t-1} \mid y_{1:t})}_{\substack{\textbf{Credibility} \\ \text{of expert}}} du_{1:t-1} \qquad (5)$$

We can think of the integral in (5) as a sum over a distribution of experts, where each expert corresponds to a single pose history, $u_{1:t-1}$. Based on its hypothesis about pose history, each expert has an *opinion* about the current pose of the object, $U_t$, and the texture maps of the object and background, $V_t$ and $B_t$. Each expert also has a *credibility*, a scalar that measures how well the expert's opinion matches the observed image $y_t$. Thus, (5) can be interpreted as follows: The filtering distribution at time $t$ is obtained by integrating over the entire ensemble of experts the opinion of each expert weighted by that expert's credibility. The opinion distribution of expert $u_{1:t-1}$ can be factorized into the expert's opinion about the pose $U_t$ times the conditional distribution of texture $V_t, B_t$ given pose:

$$\underbrace{p(u_t, v_t, b_t \mid u_{1:t-1}, y_{1:t})}_{\substack{\textbf{Opinion} \\ \text{of expert}}} = \underbrace{p(u_t \mid u_{1:t-1}, y_{1:t})}_{\textbf{Pose Opinion}} \underbrace{p(v_t, b_t \mid u_{1:t}, y_{1:t})}_{\substack{\textbf{Texture Opinion} \\ \text{given pose}}} \qquad (6)$$

The rest of this section explains how we evaluate each term in (5) and (6). We cover the distribution of texture given pose in 3.1, pose opinion in 3.2, and credibility in 3.3.

## 3.1 Texture opinion given pose

The distribution of $V_t$ and $B_t$ given the pose history $u_{1:t}$ is Gaussian with mean and covariance that can be obtained using the Kalman filter estimation equations:

$$Var^{-1}(V_t, B_t \mid u_{1:t}, y_{1:t}) = Var^{-1}(V_t, B_t \mid u_{1:t-1}, y_{1:t-1}) + a(u_t)^T \sigma_w^{-1} a(u_t) \qquad (7)$$

$$E(V_t, B_t \mid u_{1:t}, y_{1:t}) = Var(V_t, B_t \mid u_{1:t}, y_{1:t})$$
$$\times \left[ Var^{-1}(V_t, B_t \mid u_{1:t-1}, y_{1:t-1}) E(V_t, B_t \mid u_{1:t-1}, y_{1:t-1}) + a(u_t)^T \sigma_w^{-1} y_t \right] \quad (8)$$

This requires $p(V_t, B_t \mid u_{1:t-1}, y_{1:t-1})$, which we get from the Kalman prediction equations:

$$E(V_t, B_t \mid u_{1:t-1}, y_{1:t-1}) = E(V_{t-1}, B_{t-1} \mid u_{1:t-1}, y_{1:t-1}) \qquad (9)$$

$$Var(V_t, B_t \mid u_{1:t-1}, y_{1:t-1}) = Var(V_{t-1}, B_{t-1} \mid u_{1:t-1}, y_{1:t-1}) + \begin{pmatrix} \Psi_v & 0 \\ 0 & \Psi_b \end{pmatrix} \quad (10)$$

In (9), the expected value $E(V_t, B_t \mid u_{1:t-1}, y_{1:t-1})$ consists of texture maps (templates) for the object and background. In (10), $Var(V_t, B_t \mid u_{1:t-1}, y_{1:t-1})$ represents the degree of uncertainty about each texel in these texture maps. Since this is a diagonal matrix, we can refer to the mean and variance of each texel individually. For the $i$th texel in the object texture map, we use the following notation:

$$\mu_t^v(i) \stackrel{\text{def}}{=} i^{th} \text{ element of } E(V_t \mid u_{1:t-1}, y_{1:t-1})$$
$$\sigma_t^v(i) \stackrel{\text{def}}{=} (i, i)^{th} \text{ element of } Var(V_t \mid u_{1:t-1}, y_{1:t-1})$$

Similarly, define $\mu_t^b(j)$ and $\sigma_t^b(j)$ as the mean and variance of the $j$th texel in the background texture map. (This notation leaves the dependency on $u_{1:t-1}$ and $y_{1:t-1}$ implicit.)

## 3.2 Pose opinion

Based on its current texture template (derived from the history of poses and images up to time $t-1$) and the new image $y_t$, each expert $u_{1:t-1}$ has a *pose opinion*, $p(u_t \mid u_{1:t-1}, y_{1:t})$, a probability distribution representing that expert's beliefs about the pose at time $t$. Since the effect of $u_t$ on the likelihood function is nonlinear, we will not attempt to find an analytical solution for the pose opinion distribution. However, due to the spatio-temporal smoothness of video signals, it is possible to estimate the peak and variance of an expert's pose opinion.

### 3.2.1 Estimating the peak of an expert's pose opinion

We want to estimate $\hat{u}_t(u_{1:t-1})$, the value of $u_t$ that maximizes the pose opinion. Since

$$p(u_t \mid u_{1:t-1}, y_{1:t}) = \frac{p(y_{1:t-1} \mid u_{1:t-1})}{p(y_{1:t} \mid u_{1:t-1})} p(u_t \mid u_{t-1}) \, p(y_t \mid u_{1:t}, y_{1:t-1}), \qquad (11)$$

$$\hat{u}_t(u_{1:t-1}) \stackrel{\text{def}}{=} \underset{u_t}{\operatorname{argmax}} \, p(u_t \mid u_{1:t-1}, y_{1:t}) = \underset{u_t}{\operatorname{argmax}} \, p(u_t \mid u_{t-1}) \, p(y_t \mid u_{1:t}, y_{1:t-1}). \qquad (12)$$

We now need an expression for the final term in (12), the predictive distribution $p(y_t \mid u_{1:t}, y_{1:t-1})$. By integrating out the hidden texture variables from $p(y_t, v_t, b_t \mid u_{1:t}, y_{1:t-1})$, and using the conditional independence relationships defined by the graphical model (Figure 1, right), we can derive:

$$\log p(y_t \mid u_{1:t}, y_{1:t-1}) = -\frac{m}{2} \log 2\pi - \frac{1}{2} \log |Var(Y_t \mid u_{1:t}, y_{1:t-1})|$$
$$- \frac{1}{2} \sum_{i=1}^{n} \frac{(y_t(x_i(u_t)) - \mu_t^v(i))^2}{\sigma_t^v(i) + \sigma_w} - \frac{1}{2} \sum_{j \notin \mathcal{X}(u_t)} \frac{(y_t(j) - \mu_t^b(j))^2}{\sigma_t^b(j) + \sigma_w}, \qquad (13)$$

where $x_i(u_t)$ is the image pixel rendered by the $i$th object vertex when the object assumes pose $u_t$, and $\mathcal{X}(u_t)$ is the set of all image pixels rendered by the object under pose $u_t$. Combining (12) and (13), we can derive

$$\hat{u}_t(u_{1:t-1}) = \underset{u_t}{\operatorname{argmin}} \left( -\log p(u_t \mid u_{t-1}) \right. \tag{14}$$

$$\left. + \frac{1}{2} \sum_{i=1}^{n} \left[ \underbrace{\frac{[y_t(x_i(u_t)) - \mu_t^v(i)]^2}{\sigma_t^v(i) + \sigma_w}}_{\text{Foreground term}} - \underbrace{\frac{[y_t(x_i(u_t)) - \mu_t^b(x_i(u_t))]^2}{\sigma_t^b(x_i(u_t)) + \sigma_w} - \log[\sigma_t^b(x_i(u_t)) + \sigma_w]}_{\text{Background terms}} \right] \right)$$

Note the similarity between (14) and constrained optic flow (3). For example, focus on the foreground term in (14) and ignore the weights in the denominator. The previous image $y_{t-1}$ from (3) has been replaced by $\mu_t^v(\cdot)$, the estimated object texture based on the images and poses up to time $t-1$. As in optic flow, we can find the pose estimate $\hat{u}_t(u_{1:t-1})$ efficiently using the Gauss-Newton method.

### 3.2.2 Estimating the distribution of an expert's pose opinion

We estimate the distribution of an expert's pose opinion using a combination of Laplace's method and importance sampling. Suppose at time $t-1$ we are given a sample of experts indexed by $d$, each endowed with a pose sequence $u_{1:t-1}^{(d)}$, a weight $w_{t-1}^{(d)}$, and the means and variances of Gaussian distributions for object and background texture. For each expert $u_{1:t-1}^{(d)}$, we use (14) to compute $\hat{u}_t^{(d)}$, the peak of the pose distribution at time $t$ according to that expert. Define $\hat{\sigma}_t^{(d)}$ as the inverse Hessian matrix of (14) at this peak, the Laplace estimate of the covariance matrix of the expert's opinion. We then generate a set of $s$ independent samples $\{u_t^{(d,e)} : e = 1, \cdots, s\}$ from a Gaussian distribution with mean $\hat{u}_t^{(d)}$ and variance proportional to $\hat{\sigma}_t^{(d)}$, $g(\cdot | \hat{u}_t^{(d)}, \alpha\hat{\sigma}_t^{(d)})$, where the parameter $\alpha > 0$ determines the sharpness of the sampling distribution. (Note that letting $\alpha \to 0$ would be equivalent to simply setting the new pose equal to the peak of the pose opinion, $u_t^{(d,e)} = \hat{u}_t^{(d)}$.) To find the parameters of this Gaussian proposal distribution, we use the Gauss-Newton method, ignoring the second of the two background terms in (14). (This term is not ignored in the importance sampling step.)

To refine our estimate of the pose opinion we use importance sampling. We assign each sample from the proposal distribution an importance weight $w_t(d, e)$ that is proportional to the ratio between the posterior distribution and the proposal distribution:

$$\hat{p}(u_t \mid u_{1:t-1}^{(d)}, y_{1:t}) = \sum_{e=1}^{s} \delta(u_t - u_t^{(d,e)}) \frac{w_t(d, e)}{\sum_{f=1}^{s} w_t(d, f)} \tag{15}$$

$$w_t(d, e) = \frac{p(u_t^{(d,e)} \mid u_{t-1}^{(d)}) p(y_t \mid u_{1:t-1}^{(d)}, u_t^{(d,e)}, y_{1:t-1})}{g(u_t^{(d,e)} \mid \hat{u}_t^{(d)}, \alpha\hat{\sigma}_t^{(d)})} \tag{16}$$

The numerator of (16) is proportional to $p(u_t^{(d,e)} | u_{1:t-1}^{(d)}, y_{1:t})$ by (12), and the denominator of (16) is the sampling distribution.

### 3.3 Estimating an expert's credibility

The credibility of the $d$th expert, $p(u_{1:t-1}^{(d)} \mid y_{1:t})$, is proportional to the product of a prior term and a likelihood term:

$$p(u_{1:t-1}^{(d)} \mid y_{1:t}) = \frac{p(u_{1:t-1}^{(d)} \mid y_{1:t-1}) p(y_t \mid u_{1:t-1}^{(d)}, y_{1:t-1})}{p(y_t \mid y_{1:t-1})}. \tag{17}$$

Regarding the likelihood,

$$p(y_t|u_{1:t-1}, y_{1:t-1}) = \int p(y_t, u_t|u_{1:t-1}, y_{1:t-1})du_t = \int p(y_t|u_{1:t}, y_{1:t-1})p(u_t|u_{t-1})du_t \tag{18}$$

We already generated a set of samples $\{u_t^{(d,e)} : e = 1, \cdots, s\}$ that estimate the pose opinion of the $d$th expert, $p(u_t \mid u_{1:t-1}^{(d)}, y_{1:t})$. We can now use these samples to estimate the likelihood for the $d$th expert:

$$p(y_t \mid u_{1:t-1}^{(d)}, y_{1:t-1}) = \int p(y_t \mid u_{1:t-1}^{(d)}, u_t, y_{1:t-1})p(u_t \mid u_{t-1}^{(d)})du_t \tag{19}$$

$$= \int p(y_t \mid u_{1:t-1}^{(d)}, u_t, y_{1:t-1})g(u_t \mid \hat{u}_t^{(d)}, \alpha\hat{\sigma}_t^{(d)}) \frac{p(u_t \mid u_{t-1}^{(d)})}{g(u_t \mid \hat{u}_t^{(d)}, \alpha\hat{\sigma}_t^{(d)})}du_t \approx \frac{\sum_{e=1}^s w_t(d, e)}{s}$$

### 3.4 Updating the filtering distribution

Once we have calculated the opinion and credibility of each expert $u_{1:t-1}$, we evaluate the integral in (5) as a weighted sum over experts. The credibilities of all of the experts are normalized to sum to 1. New experts $u_{1:t}$ (children) are created from the old experts $u_{1:t-1}$ (parents) by appending a pose $u_t$ to the parent's history of poses $u_{1:t-1}$. Every expert in the new generation is created as follows: One parent is chosen to sire the child. The probability of being chosen is proportional to the parent's credibility. The child's value of $u_t$ is chosen at random from its parent's pose opinion (the weighted samples described in Section 3.2.2).

## 4   Relation to Optic Flow and Template Matching

In basic template-matching, the same time-invariant texture map is used to track every frame in the video sequence. Optic flow can be thought of as template-matching with a template that is completely reset at each frame for use in the subsequent frame. In most cases, optimal inference under G-flow involves a combination of optic flow-based and template-based tracking, in which the texture template gradually evolves as new images are presented. Pure optic flow and template-matching emerge as special cases.

**Optic Flow as a Special Case**   Suppose that the pose transition probability $p(u_t \mid u_{t-1})$ is uninformative, that the background is uninformative, that every texel in the initial object texture map has equal variance, $Var(V_1) = \kappa I_n$, and that the texture transition uncertainty is very high, $\Psi_v \to diag(\infty)$. Using (7), (8), and (10), it follows that:

$$\mu_t^v(i) = [a_v(u_{t-1})]^T y_{t-1} = y_{t-1}(x_i(u_{t-1})), \tag{20}$$

i.e., the object texture map at time $t$ is determined by the pixels from image $y_{t-1}$ that according to pose $u_{t-1}$ were rendered by the object. As a result, (14) reduces to:

$$\hat{u}_t(u_{1:t-1}) = \underset{u_t}{\arg\min} \frac{1}{2} \sum_{i=1}^n \left[y_t(x_i(u_t)) - y_{t-1}(x_i(u_{t-1}))\right]^2 \tag{21}$$

which is identical to (3). Thus constrained optic flow [10, 2, 11] is simply a special case of optimal inference under G-flow, with a single expert and with sampling parameter $\alpha \to 0$.

The key assumption that $\Psi_v \to diag(\infty)$ means that the object's texture is very different in adjacent frames. However, optic flow is typically applied in situations in which the object's texture in adjacent frames is similar. The optimal solution in such situations calls not for optic flow, but for a texture map that integrates information across multiple frames.

**Template Matching as a Special Case** Suppose the initial texture map is known precisely, $Var(V_1) = 0$, and the texture transition uncertainty is very low, $\Psi_v \to \mathbf{0}$. By (7), (8), and (10), it follows that $\mu_t^v(i) = \mu_{t-1}^v(i) = \mu_1^v(i)$, i.e., the texture map does not change over time, but remains fixed at its initial value (it is a texture template). Then (14) becomes:

$$\hat{u}_t(u_{1:t-1}) = \underset{u_t}{\operatorname{argmin}} \sum_{i=1}^{n} \big[ y_t(x_i(u_t)) - \mu_1^v(i) \big]^2 \tag{22}$$

where $\mu_1^v(i)$ is the $i$th texel of the fixed texture template. This is the error function minimized by standard template-matching algorithms. The key assumption that $\Psi_v \to \mathbf{0}$ means the object's texture is constant from each frame to the next, which is rarely true in real data. G-flow provides a principled way to relax this unrealistic assumption of template methods.

**General Case** In general, if the background is uninformative, then minimizing (14) results in a weighted combination of optic flow and template matching, with the weight of each approach depending on the current level of certainty about the object template. In addition, when there is useful information in the background, G-flow infers a model of the background which is used to improve tracking.

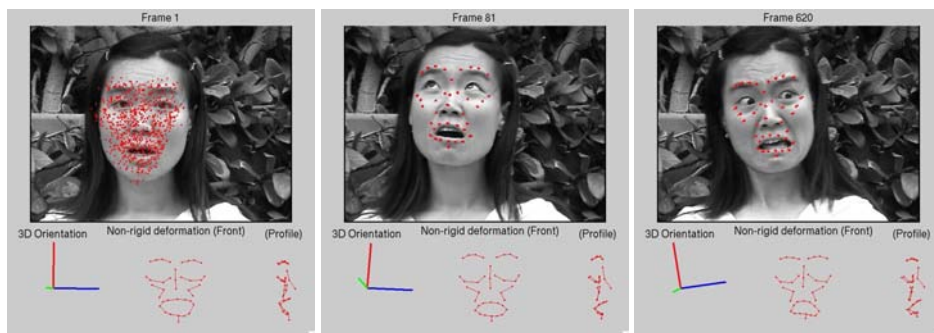

Figure 2: G-flow tracking an outdoor video. Results are shown for frames 1, 81, and 620.

## 5 Simulations

We collected a video (30 frames/sec) of a subject in an outdoor setting who made a variety of facial expressions while moving her head. A later motion-capture session was used to create a 3D morphable model of her face, consisting of a set of 5 morph bases ($k = 5$).

Twenty experts were initialized randomly near the correct pose on frame 1 of the video and propagated using G-flow inference (assuming an uninformative background). See http://mplab.ucsd.edu for video. Figure 2 shows the distribution of experts for three frames. In each frame, every expert has a hypothesis about the pose (translation, rotation, scale, and morph coefficients). The 38 points in the model are projected into the image according to each expert's pose, yielding 760 red dots in each frame. In each frame, the mean of the experts gives a single hypothesis about the 3D non-rigid deformation of the face (lower right) as well as the rigid pose of the face (rotated 3D axes, lower left). Notice G-flow's ability to recover from error: bad initial hypotheses are weeded out, leaving only good hypotheses.

To compare G-flow's performance versus deterministic constrained optic flow algorithms such as [10, 2, 11] , we used both G-flow and the method from [2] to track the same video sequence. We ran each tracker several times, introducing small errors in the starting pose.

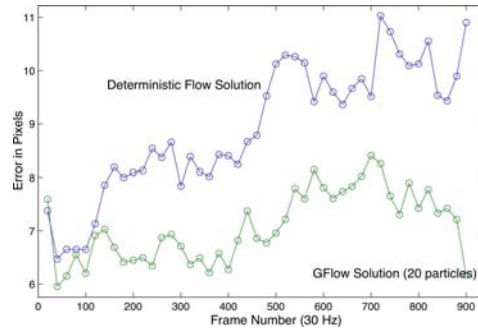

Figure 3: Average error over time for G-flow (green) and for deterministic optic flow [2] (blue). Results were averaged over 16 runs (deterministic algorithm) or 4 runs (G-flow) and smoothed.

As ground truth, the 2D locations of 6 points were hand-labeled in every 20th frame. The error at every 20th frame was calculated as the distance from these labeled locations to the inferred (tracked) locations, averaged across several runs. Figure 3 compares this tracking error as a function of time for the deterministic constrained optic flow algorithm and for a 20-expert version of the G-flow tracking algorithm. Notice that the deterministic system has a tendency to drift (increase in error) over time, whereas G-flow can recover from drift.

### Acknowledgments

Tim K. Marks was supported by NSF grant IIS-0223052 and NSF grant DGE-0333451 to GWC. John Hershey was supported by the UCDIMI grant D00-10084. J. Cooper Roddey was supported by the Swartz Foundation. Javier R. Movellan was supported by NSF grants IIS-0086107, IIS-0220141, and IIS-0223052, and by the UCDIMI grant D00-10084.

## References

[1] Simon Baker and Iain Matthews. Lucas-kanade 20 years on: A unifying framework. *International Journal of Computer Vision*, 56(3):221–255, 2002.

[2] M. Brand. Flexible flow for 3D nonrigid tracking and shape recovery. In *CVPR*, volume 1, pages 315–322, 2001.

[3] H. Chen, P. Kumar, and J. van Schuppen. On Kalman filtering for conditionally gaussian systems with random matrices. *Syst. Contr. Lett.*, 13:397–404, 1989.

[4] R. Chen and J. Liu. Mixture Kalman filters. *J. R. Statist. Soc. B*, 62:493–508, 2000.

[5] A. Doucet and C. Andrieu. Particle filtering for partially observed gaussian state space models. *J. R. Statist. Soc. B*, 64:827–838, 2002.

[6] A. Doucet, N. de Freitas, K. Murphy, and S. Russell. Rao-blackwellised particle filtering for dynamic bayesian networks. In *16th Conference on Uncertainty in AI*, pages 176–183, 2000.

[7] A. Doucet, S. J. Godsill, and C. Andrieu. On sequential monte carlo sampling methods for bayesian filtering. *Statistics and Computing*, 10:197–208, 2000.

[8] Zoubin Ghahramani and Geoffrey E. Hinton. Variational learning for switching state-space models. *Neural Computation*, 12(4):831–864, 2000.

[9] B. Lucas and T. Kanade. An iterative image registration technique with an application to stereo vision. In *Proceedings of the International Joint Conference on Artificial Intelligence*, 1981.

[10] L. Torresani, D. Yang, G. Alexander, and C. Bregler. Tracking and modeling non-rigid objects with rank constraints. In *CVPR*, pages 493–500, 2001.

[11] Lorenzo Torresani, Aaron Hertzmann, and Christoph Bregler. Learning non-rigid 3d shape from 2d motion. In *Advances in Neural Information Processing Systems 16*. MIT Press, 2004.
